# A Criterion for the Convergence of Learning with Spike Timing Dependent Plasticity

**Robert Legenstein and Wolfgang Maass**
Institute for Theoretical Computer Science
Technische Universitaet Graz
A-8010 Graz, Austria
{legi,maass}@igi.tugraz.at

## Abstract

We investigate under what conditions a neuron can learn by experimentally supported rules for spike timing dependent plasticity (STDP) to predict the arrival times of strong "teacher inputs" to the same neuron. It turns out that in contrast to the famous Perceptron Convergence Theorem, which predicts convergence of the perceptron learning rule for a simplified neuron model whenever a stable solution exists, no equally strong convergence guarantee can be given for spiking neurons with STDP. But we derive a criterion on the statistical dependency structure of input spike trains which characterizes exactly when learning with STDP will converge on average for a simple model of a spiking neuron. This criterion is reminiscent of the linear separability criterion of the Perceptron Convergence Theorem, but it applies here to the rows of a correlation matrix related to the spike inputs. In addition we show through computer simulations for more realistic neuron models that the resulting analytically predicted positive learning results not only hold for the common interpretation of STDP where STDP changes the weights of synapses, but also for a more realistic interpretation suggested by experimental data where STDP modulates the initial release probability of dynamic synapses.

## 1 Introduction

Numerous experimental data show that STDP changes the value $w_{old}$ of a synaptic weight after pairing of the firing of the presynaptic neuron at time $t^{pre}$ with a firing of the postsynaptic neuron at time $t^{post} = t^{pre} + \Delta t$ to $w_{new} = w_{old} + \Delta w$ according to the rule

$$w_{new} = \begin{cases} \min\{w_{max}, \ w_{old} + W_+ \cdot e^{-\Delta t/\tau_+}\} & , \quad \text{if } \Delta t > 0 \\ \max\{0, \ w_{old} - W_- \cdot e^{\Delta t/\tau_-}\} & , \quad \text{if } \Delta t \leq 0 \,, \end{cases} \quad (1)$$

with some parameters $W_+, W_-, \tau_+, \tau_- > 0$ (see [1]). If during training a teacher induces firing of the postsynaptic neuron, this rule becomes somewhat analogous to the well-known perceptron learning rule for McCulloch-Pitts neurons (= "perceptrons"). The Perceptron Convergence Theorem states that this rule enables a perceptron to learn, starting from any initial weights, after finitely many errors *any* transformation that it could possibly implement. However, we have constructed examples of input spike trains and teacher spike trains

(omitted in this abstract) such that although a weight vector exists which produces the desired firing and which is stable under STDP, learning with STDP does not converge to a stable solution. On the other hand experiments in vivo have shown that neurons can be taught by suitable teacher input to adopt a given firing response [2, 3] (although the spike-timing dependence is not exploited there). We show in section 2 that such convergence of learning can be explained by STDP in the average case, provided that a certain criterion is met for the statistical dependence among Poisson spike inputs. The validity of the proposed criterion is tested in section 3 for more realistic models for neurons and synapses.

## 2   An analytical criterion for the convergence of STDP

The average case analysis in this section is based on the linear Poisson neuron model (see [4, 5]). This neuron model outputs a spike train $S^{post}(t)$ which is a realization of a Poisson process with the underlying instantaneous firing rate $R^{post}(t)$. We represent a spike train $S(t)$ as a sum of Dirac-$\delta$ functions $S(t) = \sum_k \delta(t - t_k)$, where $t_k$ is the $k^{th}$ spike time of the spike train. The effect of an input spike at input $i$ at time $t'$ is modeled by an increase in the instantaneous firing rate of an amount $w_i(t')\epsilon(t - t')$, where $\epsilon$ is a response kernel and $w_i(t')$ is the synaptic efficacy of synapse $i$ at time $t'$. We assume $\epsilon(s) = 0$ for $s < 0$ (causality), $\int_0^\infty ds\, \epsilon(s) = 1$ (normalization of the response kernel), and $\epsilon(s) \geq 0$ for all $s$ as well as $w_i \geq 0$ for all $i$ (excitatory inputs). In the linear model, the contributions of all inputs are summed up linearly:

$$R^{post}(t) = \sum_{j=1}^n \int_0^\infty ds\, w_j(t - s)\, \epsilon(s)\, S_j(t - s)\,, \tag{2}$$

where $S_1, \ldots, S_n$ are the $n$ presynaptic spike trains. Note that in this spike generation process, the generation of an output spike is independent of previous output spikes.

The STDP-rule (1) avoids the growth of weights beyond bounds 0 and $w_{max}$ by simple clipping. Alternatively one can make the weight update dependent on the actual weight value. In [5] a general rule is suggested where the weight dependence has the form of a power law with a non-negative exponent $\mu$. This weight update rule is defined by

$$\Delta w = \begin{cases} W_+ \cdot (1 - w)^\mu \cdot e^{-\Delta t/\tau_+} & , \quad \text{if } \Delta t > 0 \\ -W_- \cdot w^\mu \cdot e^{\Delta t/\tau_-} & , \quad \text{if } \Delta t \leq 0 \quad , \end{cases} \tag{3}$$

where we assumed for simplicity that $w_{max} = 1$. Instead of looking at specific input spike trains, we consider the average behavior of the weight vector for (possibly correlated) homogeneous Poisson input spike trains. Hence, the change $\Delta w_i$ is a random variable with a mean drift and fluctuations around it. We will in the following focus on the drift by assuming that individual weight changes are very small and only averaged quantities enter the learning dynamics, see [6]. Let $S_i$ be the spike train of input $i$ and let $S^*$ be the output spike train of the neuron. The mean drift of synapse $i$ at time $t$ can be approximated as

$$\dot{w}_i(t) = W_+(1 - w_i)^\mu \int_0^\infty ds\, e^{-s/\tau} C_i(s;t) - W_- w_i^\mu \int_{-\infty}^0 ds\, e^{s/\tau} C_i(s;t) \quad , \tag{4}$$

where $C_i(s;t) = \langle S_i(t) S^*(t+s) \rangle_E$ is the ensemble averaged correlation function between input $i$ and the output of the neuron (see [5, 6]). For the linear Poisson neuron model, input-output correlations can be described by means of correlations in the inputs. We define the normalized cross correlation between input spike trains $S_i$ and $S_j$ with a common rate $r > 0$ as

$$C_{ij}^0(s) = \frac{\langle S_i(t)\, S_j(t+s) \rangle_E}{r^2} - 1\,, \tag{5}$$

which assumes value 0 for uncorrelated Poisson spike trains. We assume in this article that $C_{ij}^0$ is constant over time. In our setup, the output of the neuron during learning is clamped

to the teacher spike train $S^*$ which is the output of a neuron with the target weight vector $\mathbf{w}^*$. Therefore, the input-output correlations $C_i(s;t)$ are also constant over time and we denote them by $C_i(s)$ in the following. In our neuron model, correlations are shaped by the response kernel $\epsilon(s)$ and they enter the learning equation (4) with respect to the learning window. This motivates the definition of *window correlations* $c_{ij}^+$ and $c_{ij}^-$ for the positive and negative learning window respectively:

$$c_{ij}^\pm = 1 + \frac{1}{\tau} \int_0^\infty ds\, e^{-s/\tau} \int_0^\infty ds'\, \epsilon(s') C_{ij}^0(\pm s - s') \quad . \tag{6}$$

We call the matrices $C^\pm = \{c_{ij}^\pm\}_{i,j=1,\ldots,n}$ the window correlation matrices. Note that window correlations are non-negative and that for homogeneous Poisson input spike trains and for a non-negative response kernel, they are positive. For soft weight bounds and $\mu > 0$, a synaptic weight can converge to a value arbitrarily close to 0 or 1, but not to one of these values directly. This motivates the following definition of learnability.

**Definition 2.1** *We say that a target weight vector $\mathbf{w}^* \in \{0,1\}^n$ can approximately be learned in a supervised paradigm by STDP with soft weight bounds on homogeneous Poisson input spike trains (short: "$\mathbf{w}^*$ can be learned") if and only if there exist $W_+, W_- > 0$, such that for $\mu \to 0$ the ensemble averaged weight vector $\langle \mathbf{w}(t) \rangle_E$ with learning dynamics given by Equation 4 converges to $\mathbf{w}^*$ for any initial weight vector $\mathbf{w}(0) \in [0,1]^n$.*

We are now ready to formulate an analytical criterion for learnability:

**Theorem 2.1** *A weight vector $\mathbf{w}^*$ can be learned (when being teached with $S^*$) for homogeneous Poisson input spike trains with window correlation matrices $C^+$ and $C^-$ to a linear Poisson neuron with non-negative response kernel if and only if $\mathbf{w}^* \neq \mathbf{0}$ and*

$$\frac{\sum_{k=1}^n w_k^* c_{ik}^+}{\sum_{k=1}^n w_k^* c_{ik}^-} > \frac{\sum_{k=1}^n w_k^* c_{jk}^+}{\sum_{k=1}^n w_k^* c_{jk}^-}$$

*for all pairs $\langle i,j \rangle \in \{1,\ldots,n\}^2$ with $w_i^* = 1$ and $w_j^* = 0$.*

**Proof idea:** The correlation between an input and the teacher induced output is (by Eq. 2):

$$C_i(s) = \langle S_i(t)\, S^*(t+s) \rangle_E = \sum_{j=1}^n w_j^* \int_0^\infty ds'\, \epsilon(s')\, \langle S_i(t)\, S_j(t+s-s') \rangle_E .$$

Substitution of this equation into Eq. 4 yields the synaptic drift

$$\dot{w}_i = \tau r^2 \left[ W_+ (1 - w_i)^\mu \sum_{j=1}^n w_j^* c_{ij}^+ - W_- w_i^\mu \sum_{j=1}^n w_j^* c_{ij}^- \right] . \tag{7}$$

We find the equilibrium points $w_{\mu i}$ of synapse $i$ by setting $\dot{w}_i = 0$ in Eq. 7. This yields $w_{\mu i} = \left( 1 + \frac{1}{\Lambda_i^{1/\mu}} \right)^{-1}$, where $\Lambda_i$ denotes $\frac{W_+}{W_-} \frac{\sum_{j=1}^n w_j^* c_{ij}^+}{\sum_{j=1}^n w_j^* c_{ij}^-}$. Note that the drift is zero if $\mathbf{w}^* = \mathbf{0}$ which implies that $\mathbf{w}^* = \mathbf{0}$ cannot be learned. For $\mathbf{w}^* \neq \mathbf{0}$, one can show that $\mathbf{w}_\mu = (w_{\mu 1}, \ldots, w_{\mu n})$ is the only equilibrium point of the system and that it is stable. Since the system decomposes into $n$ independent one-dimensional systems, convergence to $\mathbf{w}^*$ is guaranteed for all initial conditions. Furthermore, one sees that $\lim_{\mu \to 0} w_{\mu i} = 1$ if and only if $\Lambda_i > 1$, and $\lim_{\mu \to 0} w_{\mu i} = 0$ if and only if $\Lambda_i < 1$. Therefore, $\lim_{\mu \to 0} \mathbf{w}_\mu = \mathbf{w}^*$ holds if and only if $\Lambda_i > 1$ for all $i$ with $w_i^* = 1$ and $\Lambda_i < 1$ for all $i$ with $w_i^* = 0$. The theorem follows from the definition of $\Lambda_i$. ∎

For a wide class of cross-correlation functions, one can establish a relationship between learnability by STDP and the well-known concept of linear separability from linear algebra.[1] Because of synaptic delays, the response of a spiking neuron to an input spike is delayed by some time $t_0$. One can model such a delay in the response kernel by the restriction $\epsilon(s) = 0$ for all $s \leq t_0$. In the following Corollary we consider the case where input correlations $C_{ij}^0(s)$ appear only in a time window smaller than the delay:

**Corollary 2.1** *If there exists a $t_0 \geq 0$ such that $\epsilon(s) = 0$ for all $s \leq t_0$ and $C_{ij}^0(s) = 0$ for all $s < -t_0$, $i, j \in \{1, \ldots, n\}$, then the following holds for the case of homogeneous Poisson input spike trains to a linear Poisson neuron with positive response kernel $\epsilon$:*

*A weight vector $\mathbf{w}^*$ can be learned if and only if $\mathbf{w}^* \neq \mathbf{0}$ and $\mathbf{w}^*$ linearly separates the list $L = \langle\langle \mathbf{c}_1^+, w_1^* \rangle, \ldots, \langle \mathbf{c}_n^+, w_n^* \rangle\rangle$, where $\mathbf{c}_1^+, \ldots, \mathbf{c}_n^+$ are the rows of $C^+$.*

**Proof idea:** From the assumptions of the corollary it follows that $c_{ij}^- = 1$. In this case, the condition in Theorem 2.1 is equivalent to the statement that $\mathbf{w}^*$ linearly separates the list $L = \langle\langle \mathbf{c}_1^+, w_1^* \rangle, \ldots, \langle \mathbf{c}_n^+, w_n^* \rangle\rangle$. ∎

Corollary 2.1 can be viewed as an analogon of the Perceptron Convergence Theorem for the average case analysis of STDP. Its formulation is tight in the sense that linear separability of the list $L$ alone (as opposed to linear separability by the target vector $\mathbf{w}^*$) is not sufficient to imply learnability. For uncorrelated input spike trains of rate $r > 0$, the normalized cross correlation functions are given by $C_{ij}^0(s) = \frac{\delta_{ij}}{r}\delta(s)$, where $\delta_{ij}$ is the Kronecker delta function. The positive window correlation matrix $C^+$ is therefore essentially a scaled version of the identity matrix. The following corollary then follows from Corollary 2.1:

**Corollary 2.2** *A target weight vector $\mathbf{w}^* \in \{0, 1\}^n$ can be learned in the case of uncorrelated Poisson input spike trains to a linear Poisson neuron with positive response kernel $\epsilon$ such that $\epsilon(s) = 0$ for all $s \leq 0$ if and only if $\mathbf{w}^* \neq \mathbf{0}$.*

# 3   Computer simulations of supervised learning with STDP

In order to make a theoretical analysis feasible, we needed to make in section 2 a number of simplifying assumptions on the neuron model and the synapse model. In addition a number of approximations had to be used in order to simplify the estimates. We consider in this section the more realistic integrate-and-fire model[2] for neurons and a model for synapses which are subject to paired-pulse depression and paired-pulse facilitation, in addition to the long term plasticity induced by STDP [7]. This model describes synapses with parameters $U$ (initial release probability), $D$ (depression time constant), and $F$ (facilitation time constant) in addition to the synaptic weight $w$. The parameters $U$, $D$, and $F$ were randomly

chosen from Gaussian distributions that were based on empirically found data for such connections. We also show that in some cases a less restrictive teacher forcing suffices, that tolerates undesired firing of the neuron during training. The results of section 2 predict that the temporal structure of correlations has a strong influence on the outcome of a learning experiment. We used input spike trains with cross correlations that decay exponentially with a correlation decay constant $\tau_{cc}$.[3] In experiment 1 we consider temporal correlations with $\tau_{cc}$=10ms. Since such "broader" correlations are not problematic for STDP, sharper correlations ($\tau_{cc}$=6ms) are considered in experiment 2.

**Experiment 1 (correlated input with $\tau_{cc}$=10ms):** In this experiment, a leaky integrate-and-fire neuron received inputs from 100 dynamic synapses. 90% of these synapses were excitatory and 10% were inhibitory. For each excitatory synapse, the maximal efficacy $w_{max}$ was chosen from a Gaussian distribution with mean 54 and SD 10.8, bounded by $54 \pm 3SD$. The 90 excitatory inputs were divided into 9 groups of 10 synapses per group. Spike trains were correlated within groups with correlation coefficients between 0 and 0.8, whereas there were virtually no correlations between spike trains of different groups.[4] Target weight vectors $\mathbf{w}^*$ were chosen in the most adverse way: half of the weights of $\mathbf{w}^*$ within each group was set to 0, the other half to its maximal value $w_{max}$ (see Fig. 1C).

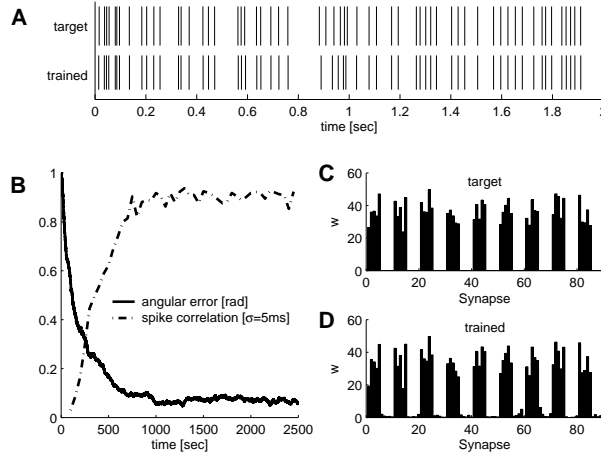

Figure 1: Learning a target weight vector $\mathbf{w}^*$ on correlated Poisson inputs. **A)** Output spike train on test data after one hour of training (trained) compared to the target output (target). **B)** Evolution of the angle between weight vector $\mathbf{w}(t)$ and the vector $\mathbf{w}^*$ that implements $F$ in radiant (angular error, solid line), and spike correlation (dashed line). **C)** Target weight vector $\mathbf{w}^*$ consisting of elements with value 0 or the value $w_{max}$ assigned to that synapse. **D)** Corresponding weights of the learned vector $\mathbf{w}(t)$ after 40 minutes of training. (All time data refer to simulated biological time)

Before training, the weights of all excitatory synapses were initialized by randomly chosen small values. Weights of inhibitory synapses remained fixed throughout the experiment. Information about the target weight vector $\mathbf{w}^*$ was given to the neuron only in the form of short current injections (1 $\mu$A for 0.2 ms) at those times when the neuron with the weight vector $\mathbf{w}^*$ would have produced a spike. Learning was implemented as standard STDP (see rule 1) with parameters $\tau_+ = \tau_- = 20ms$, $W_+ = 0.45$, $W_-/W_+ = 1.05$. Additional inhibitory input was given to the neuron during training that reduced the occurrence of non-

teacher-induced firing of the neuron (see text below).[5] Two different performance measures were used for analyzing the learning progress. The "spike correlation" measures for test inputs that were not used for training (but had been generated by the same process) the deviation between the output spike train produced by the target weight vector $\mathbf{w}^*$ for this input, and the output spike train produced for the same input by the neuron with the current weight vector $\mathbf{w}(t)$[6]. The *angular error* measures the angle between the current weight vector $\mathbf{w}(t)$ and the target weight vector $\mathbf{w}^*$. The results are shown in Fig. 1. One can see that the deviation of the learned weight vector shown in panel D from the target weight vector $\mathbf{w}^*$ (panel C) is very small, even for highly correlated groups of synapses with heterogeneous target weights. No significant changes in the results were observed for longer simulations (4 hours simulated biological time), showing stability of learning. On 20 trials (each with a new random distribution of maximal weights $w_{max}$, different initializations $\mathbf{w}(0)$ of the weight vector before learning, and new Poisson spike trains), a spike correlation of $0.83 \pm 0.06$ was achieved (angular error $6.8 \pm 4.7$ degrees). Note that learning is not only based on teacher spikes but also on non teacher-induced firing. Therefore, strongly correlated groups of inputs tend to cause autonomous (i.e., not teacher-induced) firing of the neuron which results in weight increases for *all* weights within the corresponding group of synapses according to well-known results for STDP [8, 5]. Obviously this effect makes it quite hard to learn a target weight vector $\mathbf{w}^*$ where half of the weights for each correlated group have value 0. The effect is reduced by the additional inhibitory input during training which reduces undesired firing. However, without this input a spike correlation of $0.79 \pm 0.09$ could still be achieved (angular error $14.1 \pm 10$ degrees).

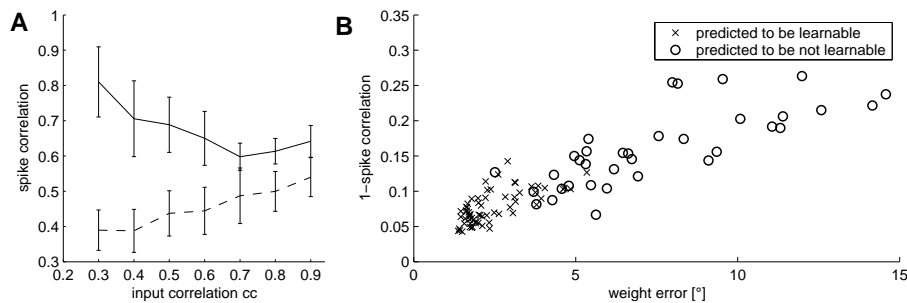

Figure 2: **A)** Spike correlation achieved for correlated inputs (solid line). Some inputs were correlated with $cc$ plotted on the x-axis. Also, as a control the spike correlation achieved by randomly drawn weight vectors is shown (dashed line, where half of the weights were set to $w_{max}$ and the other weights were set to 0). **B)** Comparison between theory and simulation results for a leaky integrate-and-fire neuron and input correlations between 0.1 and 0.5 ($\tau_{cc} = 6ms$). Each cross (open circle) marks a trial where the target vector was learnable (not learnable) according to Theorem 2.1. The actual learning performance of STDP is plotted for each trial in terms of the weight error (x-axis) and 1 minus the spike correlation (y-axis).

**Experiment 2 (testing the theoretical predictions for $\tau_{cc}$=6ms):** In order to evaluate the dependence of correlation among inputs we proceeded in a setup similar to experiment 1. 4 input groups consisting each of 10 input spike trains were constructed for which the correlations within each group had the same value $cc$ while the input spike train to the other 50 excitatory synapses were uncorrelated. Again, half of the weights of $\mathbf{w}^*$ within

each correlated group (and within the uncorrelated group) was set to 0, the other half to a randomly chosen maximal value. The learning performance after 1 hour of training for 20 trials is plotted in Fig. 2A for 7 different values of the correlation $cc$ ($\tau_{cc} = 6$ms) that is applied in 4 of the input groups (solid line).

In order to test the approximate validity of Theorem 2.1 for leaky integrate-and-fire neurons and dynamic synapses, we repeated the above experiment for input correlations $cc = 0.1, 0.2, 0.3, 0.4,$ and $0.5$. For each correlation value, 20 learning trials (with different target vectors) were simulated. For each trial we first checked whether the (randomly chosen) target vector $\mathbf{w}^*$ was learnable according to the condition given in Theorem 2.1 (65% of the 100 learning trials were classified as being learnable).[7] The actual performance of learning with STDP was evaluated after 50 minutes of training.[8] The result is shown in Fig. 2B. It shows that the theoretical prediction of learnability or non-learnability for the case of simpler neuron models and synapses from Theorem 2.1 translates in a biologically more realistic scenario into a quantitative grading of the learning performance that can ultimately be achieved with STDP.

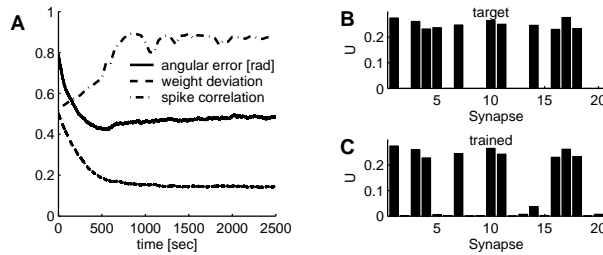

Figure 3: Results of modulation of initial release probabilities $U$. **A)** Performance of $U$-learning for a generic learning task (see text). **B)** Twenty values of the target $U$ vector (each component assumes its maximal possible value or the value 0). **C)** Corresponding $U$ values after 42 minutes of training.

**Experiment 3 (Modulation of initial release probabilities $U$ by STDP):** Experimental data from [9] suggest that synaptic plasticity does not change the uniform scaling of the amplitudes of EPSPs resulting from a presynaptic spike train (i.e., the parameter $w$), but rather redistributes the sum of their amplitudes. If one assumes that STDP changes the parameter $U$ that determines the synaptic release probability for the first spike in a spike train, whereas the weight $w$ remains unchanged, then the same experimental data that support the classical rule for STDP, support the following rule for changing $U$:

$$U_{new} = \begin{cases} min\{U_{max}, U_{old} + U_+ \cdot e^{-\Delta t/\tau_+}\} & , & \text{if } \Delta t > 0 \\ max\{0, U_{old} - U_- \cdot e^{\Delta t/\tau_-}\} & , & \text{if } \Delta t \le 0 \ , \end{cases} \tag{8}$$

with suitable nonnegative parameters $U_{max}, U_+, U_-, \tau_+, \tau_-$.

Fig. 3 shows results of an experiment where $U$ was modulated with rule (8) (similar to experiment 1, but with uncorrelated inputs). 20 repetitions of this experiment yielded after 42 minutes of training the following results: spike correlation $0.88 \pm 0.036$, angular error $27.9 \pm 3.7$ degrees, for $U_+ = 0.0012$, $U_-/U_+ = 1.055$. Apparently the output spike train is less sensitive to changes in the values of $U$ than to changes in $w$. Consequently, since

only the behavior of a neuron with vector $\mathbf{U}^*$ but not the vector $\mathbf{U}^*$ is made available to the neuron during training, the resulting correlation between target- and actual output spike trains is quite high, whereas angular error between $\mathbf{U}^*$ and $\mathbf{U}(t)$, as well as the average deviation in $U$, remain rather large.

We also repeated experiment 1 (correlated Poisson inputs) with rule (8) for $U$-learning. 20 repetitions with different target weights and different initial conditions yielded after 35 minutes of training: spike correlation $0.75 \pm 0.08$, angular error $39.3 \pm 4.8$ degrees, for $U_+ = 8 \cdot 10^{-4}, U_-/U_+ = 1.09$.

## 4  Discussion

The main conclusion of this article is that for many common distributions of input spikes a spiking neuron can learn with STDP and teacher-induced input currents any map from input spike trains to output spike trains that it could possibly implement in a stable manner.

We have shown in section 2 that a mathematical average case analysis can be carried out for supervised learning with STDP. This theoretical analysis produces the first criterion that allows us to predict whether supervised learning with STDP will succeed in spite of correlations among Poisson input spike trains. For the special case of "sharp correlations" (i.e. when the cross correlations vanish for time shifts larger than the synaptic delay) this criterion can be formulated in terms of linear separability of the rows of a correlation matrix related to the spike input, and its mathematical form is therefore reminiscent of the well-known condition for learnability in the case of perceptron learning. In this sense Corollary 2.1 can be viewed as an analogon of the Perceptron Convergence Theorem for spiking neurons with STDP.

Furthermore we have shown that an alternative interpretation of STDP where one assumes that it modulates the initial release probabilities $U$ of dynamic synapses, rather than their scaling factors $w$, gives rise to very satisfactory convergence results for learning.

**Acknowledgment:** We would like to thank Yves Fregnac, Wulfram Gerstner, and especially Henry Markram for inspiring discussions.

## Footnotes

[1] Let $\mathbf{c}_1, \ldots, \mathbf{c}_m \in \mathbb{R}^n$ and $y_1, \ldots, y_m \in \{0, 1\}$. We say that a vector $\mathbf{w} \in \mathbb{R}^n$ linearly separates the list $\langle\langle \mathbf{c}_1, y_1 \rangle, \ldots, \langle \mathbf{c}_m, y_m \rangle\rangle$ if there exists a threshold $\Theta$ such that $y_i = \text{sign}(\mathbf{c}_i \cdot \mathbf{w} - \Theta)$ for $i = 1, \ldots, m$. We define $\text{sign}(z) = 1$ if $z \geq 0$ and $\text{sign}(z) = 0$ otherwise.

[2] The membrane potential $V_m$ of the neuron is given by $\tau_m \frac{dV_m}{dt} = -(V_m - V_{resting}) + R_m \cdot (I_{syn}(t) + I_{background} + I_{inject}(t))$ where $\tau_m = C_m \cdot R_m = 30ms$ is the membrane time constant, $R_m = 1M\Omega$ is the membrane resistance, $I_{syn}(t)$ is the current supplied by the synapses, $I_{background}$ is a constant background current, and $I_{inject}(t)$ represents currents induced by a 'teacher'. If $V_m$ exceeds the threshold voltage $V_{thresh}$ it is reset to $V_{reset} = 14.2mV$ and held there for the length $T_{refract} = 3ms$ of the absolute refractory period. *Neuron parameters:* $V_{resting} = 0V$, $I_{background}$ randomly chosen for each trial from the interval $[13.5nA, 14.5nA]$. $V_{thresh}$ was set such that each neuron spiked at a rate of about 25 Hz. This resulted in a threshold voltage slightly above $15mV$. *Synaptic parameters:* Synaptic currents were modeled as exponentially decaying currents with decay time constants $\tau_S = 3ms$ ($\tau_S = 6ms$) for excitatory (inhibitory) synapses.

[3]We constructed input spike trains with normalized cross correlations (see Equation 5) approximately given by $C^0_{ij}(s) = \frac{cc_{ij}}{2\tau_{cc}r}e^{-|s|/\tau_{cc}}$ between inputs $i$ and $j$ for a mean input rate of $r = 20$Hz, a correlation coefficient $c_{ij}$, and a correlation decay constant of $\tau_{cc} = 10$ms.

[4]The correlation coefficient $c_{ij}$ for spike trains within group $k$ consisting of 10 spike trains was set to $c_{ij} = cc_k = 0.1 * (k-1)$ for $k = 1, \ldots, 9$.

[5]We added 30 inhibitory synapses with weights drawn from a gamma distribution with mean 25 and standard deviation 7.5, that received additional 30 uncorrelated Poisson spike trains at 20 Hz.

[6]For that purpose each spike in these two output spike trains was replaced by a Gaussian function with an SD of 5 ms. The *spike correlation* between both output spike trains was defined as the correlation between the resulting smooth functions of time (for segments of length 100 s).

[7]We had chosen a response kernel of the form $\epsilon(s) = \frac{1}{\tau_1 - \tau_2}(e^{-s/\tau_1} - e^{-s/\tau_2})$ with $\tau_1 = 2ms$ and $\tau_2 = 1ms$ (Least mean squares fit of the double exponential to the peri-stimulus-time histogram (PSTH) of the neuron, which reflects the probability of spiking as a function of time $s$ since an input spike), and calculated the window correlations $c_{ij}^+$ and $c_{ij}^-$ numerically.

[8]To guarantee the best possible performance for each learning trial, training was performed on 27 different values for $W_-/W_+$ between 1.02 and 1.15.

## References

[1] L. F. Abbott and S. B. Nelson. Synaptic plasticity: taming the beast. *Nature Neurosci.*, 3:1178–1183, 2000.

[2] Y. Fregnac, D. Shulz, S. Thorpe, and E. Bienenstock. A cellular analogue of visual cortical plasticity. *Nature*, 333(6171):367–370, 1988.

[3] D. Debanne, D. E. Shulz, and Y. Fregnac. Activity dependent regulation of on- and off-responses in cat visual cortical receptive fields. *Journal of Physiology*, 508:523–548, 1998.

[4] R. Kempter, W. Gerstner, and J. L. van Hemmen. Intrinsic stabilization of output rates by spike-based hebbian learning. *Neural Computation*, 13:2709–2741, 2001.

[5] R. Gütig, R. Aharonov, S. Rotter, and H. Sompolinsky. Learning input correlations through non-linear temporally asymmetric hebbian plasticity. *Journal of Neurosci.*, 23:3697–3714, 2003.

[6] R. Kempter, W. Gerstner, and J. L. van Hemmen. Hebbian learning and spiking neurons. *Phys. Rev. E*, 59(4):4498–4514, 1999.

[7] H. Markram, Y. Wang, and M. Tsodyks. Differential signaling via the same axon of neocortical pyramidal neurons. *PNAS*, 95:5323–5328, 1998.

[8] S. Song, K. D. Miller, and L. F. Abbott. Competitive hebbian learning through spike-timing dependent synaptic plasticity. *Nature Neuroscience*, 3:919–926, 2000.

[9] H. Markram and M. Tsodyks. Redistribution of synaptic efficacy between neocortical pyramidal neurons. *Nature*, 382:807–810, 1996.
